# SpikedAttention: Training-Free and Fully Spike-Driven Transformer-to-SNN Conversion with Winner-Oriented Spike Shift for Softmax Operation

**Sangwoo Hwang**
Electrical Engineering and Computer Science
Daegu Gyeongbuk Institute of Science and Technology (DGIST)
nemesis0523@dgist.ac.kr

**Seunghyun Lee**
School of Electrical Engineering
Korea University
coder@korea.ac.kr

**Dahoon Park**
School of Electrical Engineering
Korea University
manyteacher93@korea.ac.kr

**Donghun Lee**
School of Electrical Engineering
Korea University
dhleeids@korea.ac.kr

**Jaeha Kung**[*]
School of Electrical Engineering
Korea University
jhkung@korea.ac.kr

## Abstract

Event-driven spiking neural networks (SNNs) are promising neural networks that reduce the energy consumption of continuously growing AI models. Recently, keeping pace with the development of transformers, transformer-based SNNs were presented. Due to the incompatibility of self-attention with spikes, however, existing transformer-based SNNs limit themselves by either restructuring self-attention architecture or conforming to non-spike computations. In this work, we propose a novel transformer-to-SNN conversion method that outputs an end-to-end spike-based transformer, named SpikedAttention. Our method directly converts the well-trained transformer without modifying its attention architecture. For the vision task, the proposed method converts Swin Transformer into an SNN without post-training or conversion-aware training, achieving state-of-the-art SNN accuracy on ImageNet dataset, i.e., $80.0\%$ with 28.7M parameters. Considering weight accumulation, neuron potential update, and on-chip data movement, SpikedAttention reduces energy consumption by 42% compared to the baseline ANN, i.e., Swin-T. Furthermore, for the first time, we demonstrate that SpikedAttention successfully converts a BERT model to an SNN with only 0.3% accuracy loss on average consuming 58% less energy on GLUE benchmark. Our code is available at Github ( https://github.com/sangwoohwang/SpikedAttention ).

## 1 Introduction

Recently, many practical AI applications such as conversational question answering (1), intelligent code completion (2), weather forecasting (3), and text-to-image generation (4) have rapidly evolved with the advances in artificial neural networks (ANNs) based on transformer architecture. Transform-

---

[*]*J. Kung is the corresponding author.*

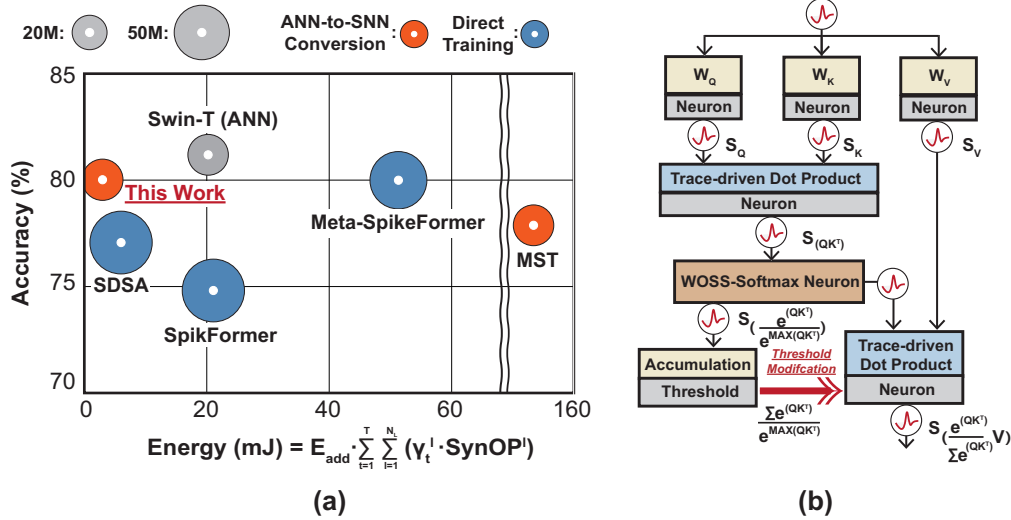

Figure 1: (a) Accuracy, energy consumption (refer to Appendix A.1), and parameter size of the propsoed SpikedAttention and other spike-based SNNs on ImageNet classification (15). (b) The structure of the proposed fully spike-based attention module (more details are in Section 4).

ers are powered by the self-attention mechanism, which extracts long-range dependencies between input tokens, e.g., words (5), image patches (6), or both (7). However, deploying state-of-the-art transformers requires a large amount of computations and a huge memory footprint. For instance, generating a single token on a GPT3-175B model requires at least 350 GFLOPs (8).

In order to reduce the ever-growing computational overhead of transformers, there were some efforts to improve their energy efficiency by utilizing spiking neural networks (SNNs). The major strength of an SNN, which mimics a biological neuron, is that only weight accumulations (ACs) are required when input spikes are encountered (*event-driven*), instead of power-hungry multiply-accumulate (MAC) operations in ANNs. The energy consumption of a 32-bit MAC is $5\times$ higher than that of a 32-bit accumulation (9). To exploit the high energy efficiency of SNNs, many CNN models based on SNNs (10; 11) have been proposed, and recently, SNN-based transformers have emerged. Implementing an SNN-based transformer is done either by direct training via a surrogate gradient (12; 13) or converting a well-trained transformer to an SNN (14).

When implementing the attention module in a spike-based transformer, there are two types of matrix multiplication. One is associated with generating $Q$, $K$, and $V$ where *static* weight matrices ($W_Q$, $W_K$, and $W_V$) are used. In this case, when an input neuron fires, its associated weight value gets accumulated to the output neuron's potential. Another type is the multiplication between *dynamically* generated matrices, e.g., $Q \cdot K^T$. In previous works (12; 14), logical AND operations are performed between two "rate-coded" spike trains, e.g., $S_{Q(i,k)}$ and $S_{K(j,k)}$, for each dot product. However, due to the probabilistic nature of the rate coding, it requires a long timestep ($T > 128$) to maintain high ANN-to-SNN conversion accuracy (14). Unfortunately, this translates to high energy consumption in running SNNs (refer to MST in Fig. 1).

Another challenge with realizing spike-based transformers is the softmax operation. In the field of ANN quantization, there have been many efforts to approximate softmax operations (16; 17). I-BERT (16) approximates the exponential function with a second-order polynomial, successfully converting it to an integer operation (*i-exp*), and FQ-ViT (17) combines *i-exp* with logarithmic quantization. In SNN-based transformers, however, softmax has not been converted to a spike-based operation due to the presence of exponential functions. Since SNNs have binary inputs and outputs, previous approaches in ANN quantization cannot be directly applied.

**Our Contribution:** In this work, we present an end-to-end spike-based transformer, named *SpikedAttention*, solving the abovementioned challenges to achieve state-of-the-art accuracy (0.6% accuracy loss compared to ANN) among spike-based transformers (Fig. 1(a)). In addition, it consumes the minimum energy, which is 42% lower than the original Swin Transformer (6), for image classification task. Furthermore, for the first time, we demonstrate that a BERT model can be successfully converted

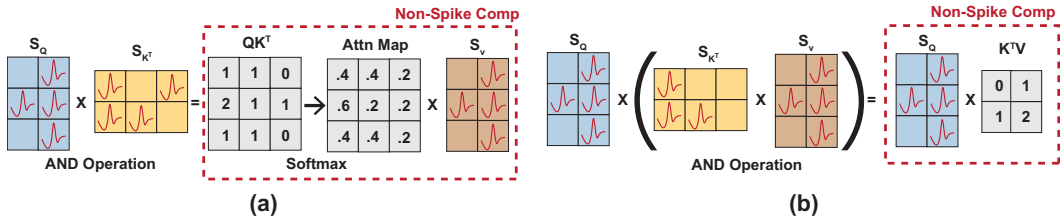

Figure 2: The computation of attention in previous SNN-based transformers (a) with the softmax (14), and (b) without the softmax operation (12). Both involve non-spike computations. Total timestep $T$ of each spike tensor is set to 1 for simplicity.

to an SNN, i.e., SpikedAttention, with negligible accuracy loss (0.3% on average) while consuming 58% less energy on GLUE benchmark. Note that the SpikedAttention does not require any additional training since it is created by the direct transformer-to-SNN conversion method thanks to the proposed fully spike-based attention module (Fig. 1(b)). The main contributions are summarized below:

- **Fully spike-based transformer**: SpikedAttention encodes external inputs and all intermediate features in spikes. Especially, each element in intermediate features is encoded as a single spike for a given timestep $T$ to minimize the energy consumption.

- **Trace-driven matrix multiplication**: To reduce the required timestep $T$ in performing multiplication between two dynamically generated matrices, we propose a trace-driven matrix multiplication. Owing to the reduced timestep, we can reduce energy consumption.

- **Exponent-free spike-based softmax**: We present a winner-oriented spike shift that controls the output spike timing to realize the spike-based softmax. This allows us to keep the vanilla self-attention architecture of the original transformer to maintain high accuracy.

## 2 Related Work

Most SNN studies have focused on replacing CNNs to improve energy efficiency in vision tasks (10; 18). With the recent advent of transformers, there have been increasing efforts to convert transformers into SNNs (14) or directly train spike-based transformers (12; 13; 19; 20). When realizing a spike-based transformer, self-attention is the major hurdle in converting the transformer to a *fully spike-driven* network. Here, "fully spike-driven" means that no real-valued multiplications or nonlinear functions are involved in any operations. Fig. 2(a) illustrates the required operations in vanilla self-attention (VSA) (5) but with $Q$, $K$, and $V$ in the form of spikes (14).

In the original VSA module, real-valued multiplications ($Q \cdot K^T$ and $Attn \cdot V$) and a softmax operation ($Attn = softmax(Q \cdot K^T / \sqrt{d})$) are required. In MST (14), the authors encode $Q$, $K$, and $V$ as spike trains (denoted by $S_Q$, $S_K$, and $S_V$ in Fig. 2(a)). By doing so, the matrix multiplication ($Q \cdot K^T$) simply becomes AND operations followed by floating-point accumulations. To convey meaningful information (in the form of spikes) to the next layer, however, a long timestep $T \geq 128$ is required in (14). This is due to the fact that the simultaneous firing of spikes at both $S_Q$ and $S_{K^T}$ is rare. This problem gets worse for temporal coding or phase coding, which produces much fewer spikes than the rate coding. Moreover, in MST, the complex softmax operations are done to compute attention maps, which requires non-spike computations (i.e., expensive exponential functions).

To remove the burden of computing the softmax, some previous studies on direct training of SNNs for transformers have restructured the attention module (12; 13). As shown in Fig. 2(b), Spikformer (12) removes the softmax operation since '$S_Q \cdot S_{K^T}$' or '$S_{K^T} \cdot S_V$' only produces non-negative values. There was another attempt to modify the attention module, such as replacing '$S_Q \cdot S_{K^T}$' with a Hadamard product followed by the column summation to obtain a mask vector that masks out less important channels in $S_V$ (13). Since these prior works restructure the network topology, they need to train the model from scratch with surrogate gradients to improve the performance.

# 3 Preliminaries

## 3.1 Neuron Model

There has been a long-standing effort in neuroscience to mathematically model biological neurons (21; 22). Neuron models with high biological plausibility are computationally expensive, while models with low biological similarity are energy efficient. Due to a high volume of parameters and significant computations involved in ANNs, energy efficiency has become more of a priority than biological plausibility when replacing them with SNN counterparts. Therefore, the most energy efficient neuron model, i.e., the leaky integrated-and-fire (LIF) model (23), is widely used. The LIF neuron model is defined as

$$v_j(t) = \lambda v_j(t-1) + \sum_{i=1}^{N_{pre}} w_{ij} S_i(t),$$

$$S_j(t) = 1, \quad v_j(t) = v_j(t) - V_\theta \quad \text{when} \quad v_j(t) \geq V_\theta, \tag{1}$$

where $i$ or $j$ is the index of a pre- or post-synaptic neuron, $v_j$ is the potential of the neuron $j$, and $\lambda$ is the leak factor of potential that causes the $v_j$ to gradually decay as the time $t$ proceeds. The $w_{ij}$ is the synaptic weight between the neuron $i$ and $j$ converted from a pre-trained ANN or directly trained. The $S_i(t)$ represents the binary spike (0 or 1) from the neuron $i$ at time $t$. The $N_{pre}$ is the number of pre-synaptic neurons. The pre-synaptic neurons that generate spikes activate the weight accumulation on $v_j(t)$. When the $v_j(t)$ exceeds the threshold $V_\theta$, the neuron $j$ fires a spike ($S_j(t) = 1$) and its potential resets to $v_j(t) - V_\theta$. Note that decreasing the potential by a threshold instead of resetting it to zero is a common technique in ANN-to-SNN conversion methods (10; 24).

## 3.2 Spike Coding Schemes

To achieve higher SNN performance, it is essential to determine how real-valued inputs are encoded into a sequence of spikes, known as 'spike coding'. The spike coding is a scheme that determines whether to generate a spike at time $t$ when converting continuous values into a set of spikes. The most widely used coding scheme is rate coding, which generates multiple spikes at each neuron via Poisson sampling over pre-defined timestep $T$ (10; 25). The rate coding generates more spikes as it converts larger values into spikes. However, increasing the number of spikes and the timestep for higher resolution leads to higher energy consumption due to more weight accumulations in Eq. (1).

Unlike the rate coding, which generates multiple spikes at each neuron, temporal coding generates a single spike per neuron. In the temporal coding, each spike represents a different value at a different spike time $t$ for a given total timestep $T$. The spike at an earlier time represents a larger value. For instance, an earlier spike can represent a linearly larger value as $\frac{T-t}{T}$ (11; 26) or an exponentially larger value as $e^{\frac{T-t}{T}}$ (27). The temporal coding is energy efficient but requires a long timestep $T$ to increase the precision/resolution.

The alternative coding scheme, known as phase coding (28; 29), combines the characteristics of the temporal coding and the rate coding. Like the temporal coding, the phase coding assigns each timestep $t$ a different value, i.e., $B^{-t}$, where $B$ is the base of the phase. As proven in Appendix B, the phase coding can be realized by replacing the leak factor ($\lambda$) of the widely used LIF neuron model in Eq. (1) with $B$, making the spike at one timestep earlier has the $B\times$ larger impact on the potential increase. When $B = 2$, we call it a binary coding. Unlike the temporal coding, the phase coding allows multiple spikes when encoding a value. For a fixed (total) timestep $T$, the phase coding provides a higher resolution than other coding schemes. Recently proposed one-spike phase coding maximizes the energy efficiency, where the sequence of spikes is approximated by the first spike (30). Note that the neuron ceases to update its potential after the first spike being generated because it is unnecessary to fire the following spikes at the cost of approximation error. In the one-spike phase coding, $V_\theta$ is tuned to the midpoint between two consecutive phase-based weights, i.e., '$(B^{-t} + B^{-t-1})/2$', to reduce the approximation error. The authors in (30) have proposed a technique to reduce the approximation error by decreasing the base $B$ of phase coding. However, decreasing the base is only guaranteed to reduce the error when $T$ is large enough. In SpikedAttention, therefore, we use the one-spike phase coding by carefully selecting the proper base $B$ and total timestep $T$ to balance well between the accuracy and the energy efficiency. The details on selecting $B$ and $T$ are presented in Appendix E.

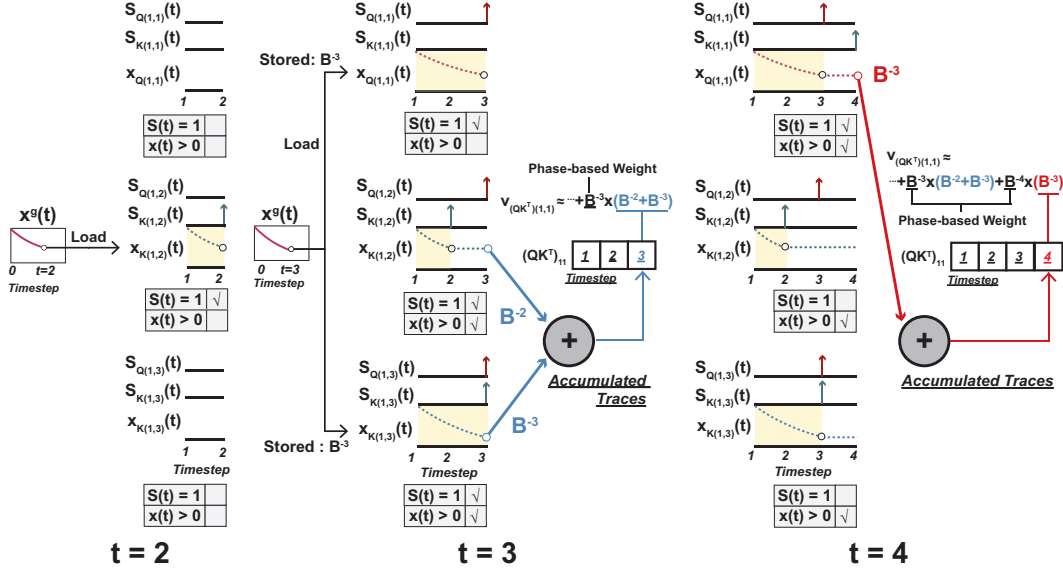

Figure 3: The proposed trace-driven dot product in SpikedAttention. A global trace decays by its base ($B$) at each timestep and the trace is transferred to each neuron's local memory when its first spike, from either $S_Q$ or $S_K$, is observed. The $v_{(QK^T)(1,1)}$ is the potential of the neuron $(QK^T)_{1,1}$.

## 4 Fully Spike-based Attention: SpikedAttention

The main goal of designing the SpikedAttention was to encode external inputs and all intermediate features by binary spikes *without altering the VSA structure* at all. Prior work on spike-based vision transformers (12; 13; 14) feed in floating-point inputs, i.e., 2D images, to the patch partition module instead of binary spikes. Also, the nonlinear nature of softmax and the high sparsity in $S_Q$, $S_K$, and $S_V$ make things more challenging in making fully spike-based transformers. In the proposed SpikedAttention, all intermediate neurons fire a single spike using the one-spike phase coding ($B < 2$). Only the external input allows multiple spikes with the binary coding, i.e., phase coding with $B = 2$. Until now, previous transformer-to-SNN conversion methods (14) could not achieve fully spike-based computations due to softmax operations, and they require a long timestep due to extremely high sparsity after AND operations between two spike-based vectors. The direct SNN training methods (12; 13) simplified the VSA structure by removing softmax, which degrades the performance. Therefore, we present two novel techniques to (i) *minimize the spike timestep* by performing trace-driven matrix multiplication and (ii) exponent-free spike-based softmax to realize the transformer-to-SNN conversion method *without any training*.

### 4.1 Trace-driven Matrix Multiplication

We focus on developing an efficient spike-based computing scheme for the multiplication between two dynamically generated spike-based matrices. We utilize a global trace that tracks the phase at time $t$ to consider the values associated with every spike. With the rate coding, probabilistic multiplications are performed with logical AND between two spike trains. However, since we use one-spike phase coding, each spike must carry its information to the next layer. Consider an example of performing $S_Q \cdot S_{K^T}$. The output value at $(i,j)$, denoted as $(QK^T)_{i,j}$, is computed by the dot product which is $\sum_k S_{Q(i,k)} S_{K(j,k)}$. Each spike train in $S_Q$ (or $S_K$) consists of only one spike which represents the value $B^{-t_{Q(i,k)}}$, where $t_{Q(i,k)}$ is the spike time of $S_{Q(i,k)}$ and $B$ is the base. Thus, the dot product output should become $\sum_k B^{-t_{Q(i,k)}} \cdot B^{-t_{K(j,k)}}$.

As mentioned earlier, our work exploits the 'spike trace model' to record the value associated with the spike that fires first among the neuron pair, e.g., $S_{Q(i,k)}$ and $S_{K(j,k)}$. In neuroscience research, spike trace is a popular method to determine the correlation between connected neurons by relative spike timing (31). Typically, the trace of a neuron '$n$', i.e., $x_n(t)$, reflects the history of spikes during timestep $T$. We simplify the trace model to globally track the value $B^{-t}$ at spike time $t$, which is

$$x^g(t) = x^g(t-1)/B, \tag{2}$$

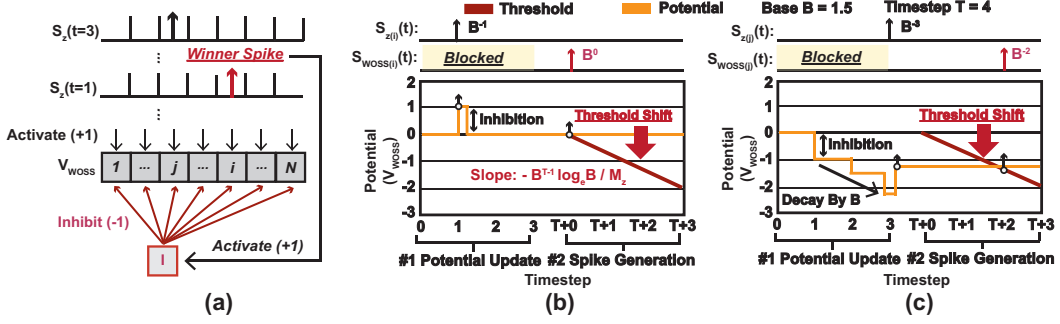

Figure 4: The proposed winner-oriented spike shift (WOSS) for approximating softmax. (a) Each WOSS neuron increases its potential ($V_{WOSS_i}$) by the incoming input spike $S_{\mathbf{z}(i)}(t)$. The first (winner) spike among $S_{\mathbf{z}}(t)$ activates a (global) inhibitory neuron which depresses all neurons. (b) Potential at neuron $i$ (winner) and its corresponding output spike at $t = T + 0$. We are shifting the first spike time from $t = 1$ to $t = 0$. The following spikes are time-shifted by the same amount. (c) Potential at neuron $j$ (non-winner) and its corresponding output spike at $t = T + 2$ (effectively, $t = 2$) due to the global inhibition and the threshold shift.

where $x^g(t)$ is the global trace which is initialized to 1 at $t = 0$. With the one-spike neuron model, when a neuron encounters the first spike at $t$, we store $B^{-t}$ in its local memory as its own trace $x_n(t)$, which will be used later to compute the dot product. We intentionally leveraged the trace-driven method since neuromorphic chips such as Loihi (32) already have the hardware module, e.g., trace memory, for updating neuron traces for widely used STDP learning.

Fig. 3 illustrates the process of a dot product for computing $(QK^T)_{1,1}$ with spike traces. If $S_{Q(1,1)}$ generates a spike earlier ($t = 3$) than $S_{K(1,1)}$, the trace value $B^{-3}$ is stored in the local memory of the neuron $S_{Q(1,1)}$. When the neuron $S_{K(1,1)}$ exceeds the threshold and generates the spike at $t = 4$, the trace of $S_{Q(1,1)}$ ($= B^{-3}$) propagates to update the potential of $(QK^T)_{1,1}$. Since the current phase is $B^{-4}$ at $t = 4$, propagation of $B^{-3}$ at $t = 4$ (colored in red) is equivalent to '$B^{-4} \times B^{-3}$'. There might be a situation in which multiple neuron pairs observe the second spike simultaneously. For instance, $S_{Q(1,2)}$-$S_{K(1,2)}$ and $S_{Q(1,3)}$-$S_{K(1,3)}$ pairs have the second spike at $t = 3$ (it may coincide with the first spike like the $S_{Q(1,3)}$-$S_{K(1,3)}$ pair). Thus, at $t = 3$, both pairs already have their stored trace for one of two neurons ($S_Q$ or $S_K$) that fired the earlier spike in the pair. Then, pre-stored traces from those pairs that share the second spike time will be accumulated together, i.e., $(B^{-2} + B^{-3})$ in this case (colored in blue). The accumulated traces are weighted by the phase of the current timestep, i.e., $B^{-3}$. By accumulating traces first, we can bound the number of multiplications to the timestep $T$ of the neuron model irrespective of the number of input neurons. Usually, an input spike tensor $S_Q$ or $S_K$ lies in $\mathbb{R}^{N \times D \times T}$ where $N$ is the number of image patches (or token length), $D$ is the embedding dimension, and $T$ is the total timestep. The accumulations happen across the dimension $D$, making the proposed approach energy efficient when $T \ll D$, which is the general case for transformers.

## 4.2 Winner-Oriented Spike Shift (WOSS) for Softmax

The main issue with the typical softmax operation is the presence of exponential functions. Our work uses the normalized softmax similar to (33), which is expressed as follows:

$$\sigma(z_i) = \frac{\exp(z_i)}{\sum_{j=0}^{N-1} \exp(z_j))} = \frac{\frac{\exp(z_i)}{\exp(\max(\mathbf{z}))}}{\sum_{j=0}^{N-1} \frac{\exp(z_j)}{\exp(\max(\mathbf{z}))}}, \quad (3)$$

where $\mathbf{z} \in \mathbb{R}^N$ and $\sigma(z_i)$ is the softmax output with respect to the $i^{\text{th}}$ neuron. Our SNN approximates $\frac{\exp(z_i)}{\exp(\max(\mathbf{z}))}$ by using the proposed WOSS neuron model and scales the threshold ($V_\theta$) of the following layer by their sum ($\sum_{j=0}^{N-1} \frac{\exp(z_j)}{\exp(\max(\mathbf{z}))}$), leading to the final softmax result. Note that multiplying the threshold by the sum term has the effect of dividing inputs by the same amount.

Since SpikedAttention generates a single spike per neuron, the objective is to approximate $\frac{\exp(z_i)}{\exp(\max(\mathbf{z}))}$ with $B^{-t_i} \in [0, 1]$. Before converting it to a spike-based representation, we extract the maximum value of $\mathbf{z}$, i.e., $M_{\mathbf{z}}$, by running a proxy training dataset. Then, by setting $z_i = M_{\mathbf{z}} z_i'$ and by applying $\log_B$ on the normalized exponential term, it can be expressed as

$$\log_B\left(\frac{\exp(M_{\mathbf{z}} z_i')}{\exp(\max(M_{\mathbf{z}} \mathbf{z}'))}\right) = \frac{(z_i' - \max(\mathbf{z}'))M_{\mathbf{z}}}{\log_e B} \approx -t_i, \tag{4}$$

where $t_i$ is the required spike time at neuron $i$ to approximate the left-hand side of Eq. (4). The $\mathbf{z}'$ arrives at WOSS neurons by spikes $S_{\mathbf{z}}$ from the previous layer, i.e., each row of $S_{(QK^T)}$, with the form of the one-spike phase coding.

First, we need to compute '$z_i' - \max(\mathbf{z}')$' from the input spikes, i.e., $S_{\mathbf{z}}$. To do so, we first find $\max(\mathbf{z}')$ by detecting the first incoming spike, called a *winner spike*. In Fig. 4(a), the winner spike is observed at neuron $i$. For the neuron $i$, its potential increases by 1 at $t = 1$ (Fig. 4(b)). Inspired by inhibitory neurons in (25), the winner spike activates the global inhibition so that potentials of all neurons (including $i$) within the same softmax group decrease by 1. The global inhibition performs the subtraction by $\max(\mathbf{z}')$ in Eq. (4). After the global inhibition at $t = 1$, non-winner neurons receive an input spike at $t > 1$. Fig. 4(c) shows how the potential of non-winner neuron $j$ changes. At $t = 3$, the neuron $j$ receives the input spike, increasing the potential by 1. At each timestep $t$, the potential decays by the base $B$ (at $t = 2$ and $t = 3$ in Fig. 4(c)), which is identical to the leak factor $\lambda$ in Eq. (1). This decay factor allows us to assign a dedicated phase $B^{-t}$ to each timestep. Thus, after the first "potential update" stage ($t \geq T$), each neuron $i$ has the fixed potential equal to

$$V_{WOSS(i)}(t \geq T) = B^{(T-1)}(z_i' - \max(\mathbf{z}')). \tag{5}$$

Now, we need to generate a single output spike at each neuron by comparing its potential to the threshold (i.e., "spike generation" stage in Fig. 4(b-c)). The threshold is initially set to 0 so that the winner-spike is time-shifted and fires at $t = 0$ for a more precise approximation of softmax, i.e., utilizing full timestep $T$ (Fig. 4(b)). The desired time $t_i$ that each neuron should fire approximately equals to $\frac{(\max(\mathbf{z}') - z_i')M_{\mathbf{z}}}{\log_e B}$. To find the $t_i$, we shift the threshold by following

$$\frac{dV_\theta(t)}{dt} = -B^{T-1}\frac{\log_e B}{M_{\mathbf{z}}}, \tag{6}$$

and generate output spike comparing the threshold with the neuron's potential. Note that the right-hand side of Eq. (6) is a constant and thus can be pre-computed. The appearance of $B^{T-1}$ is to match the scale of the potential after the "potential update" stage. The $-\frac{\log_e B}{M_{\mathbf{z}}}$ term is to consider the $M_{\mathbf{z}}/\log_e B$ in Eq. (4). Then, by comparing the Eq. (5) with the shifted threshold $V_\theta(t_i)$, we can generate a spike if the threshold becomes more negative (Fig. 4(c)) that satisfies

$$t_i > (\max(\mathbf{z}') - z_i')\frac{M_{\mathbf{z}}}{\log_e B}. \tag{7}$$

The output spike from the WOSS neuron, i.e., $S_{WOSS}$, approximates $\frac{\exp(z_i)}{\exp(\max(\mathbf{z}))}$. Thus, it needs to be divided by $\sum \frac{\exp(z_i)}{\exp(\max(\mathbf{z}))}$ to complete the softmax calculation. This is done by amplifying the threshold at the following trace-driven dot products (i.e., $S_{Attn} \cdot S_V$) by the sum term, which is the summation of spikes from the WOSS neurons in the same group, i.e., each row of $S_{WOSS}$. Similar to Eq. (1), the threshold is incremented by $S_{WOSS}(t)$ and multiplied by $B$ at each timestep until it reaches $\sum \frac{\exp(z_i)}{\exp(\max(\mathbf{z}))}$, which is expressed as follows:

$$V_{\theta(i,:)}(t+1) = B \times V_{\theta(i,:)}(t) + \sum_{j \in \mathbf{z}_i} S_{WOSS(i,j)}(t), \tag{8}$$

where $V_\theta(t = 0)$ is initially set to 1. Note that all elements at each row $i$ share the same threshold $V_{\theta(i,:)}$. Finally, the output neuron at the final trace-driven dot product in Fig. 1(b) generates spikes that approximate 'softmax$(S_Q \cdot S_{K^T}/\sqrt{d}) \cdot S_V$'. Note that implementing WOSS neurons for softmax in the neuromorphic hardware incurs 9.88% area and 12.35% power overheads compared to the hardware only with LIF neurons with no support of softmax (details are discussed in Appendix C).

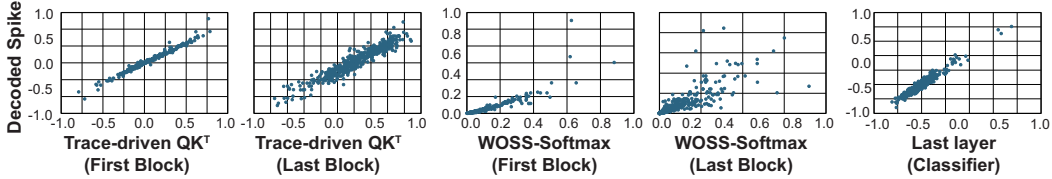

Figure 5: Scatter plots showing the correlation between the actual activation values in Swin Transformer (x-axis) and the decoded spike values in converted SpikedAttention (y-axis). The parameters are set to $T = 40, B = 1.15$.

## 5 Experimental Results

### 5.1 Conversion of Swin Transformer to SpikedAttention

The proposed SpikedAttention is implemented by SpikingJelly (34) and PyTorch computing with four NVIDIA GeForce RTX 4090 GPUs. For evaluation, Swin Transformer (6) is selected as a baseline and converted to an SNN, i.e., SpikedAttention, to perform image classification. Conventional transformers require GeLU and layer normalization, which are non-spike computations. Thus, as a pre-trained model, we replaced GeLU with ReLU and layer normalization with batch normalization according to (35). Since some activations are not followed by the ReLU function in Swin-T, those cannot be converted to unsigned/positive spikes. To cover both positive and negative values, we utilized a signed neuron (36) that generates a positive spike $(+1)$ if the potential is greater than $V_\theta$ and a negative spike $(-1)$ if the potential is less than $V_\theta$. Therefore, we trained two baselines, i.e., one with ReLUs at every layer and another with no change, to convert them to SNNs. Fig. 5 illustrates the correlation between the decoded spike values and the actual activations at trace-driven matrix multiplication and softmax layers of SpikedAttention. The earlier layers (e.g., First Block) have little conversion error, but the deeper layers (e.g., Last Block) have higher conversion error. Nevertheless, the classifier exhibits a high correlation between the actual activations and the decoded spikes.

Table 1: Comparison between SpikedAttention and the prior work in terms of the parameter size, the energy consumption, the required timestep, and the accuracy on ImageNet classification task

| Model | Method | Param (M) | Energy (mJ)[†] | Timestep | Acc (%) |
|---|---|---|---|---|---|
| Spikformer (12) | Direct Training | 66.3 | 21.5 | 4 | 74.8 |
| SDSA (13) | Direct Training | 66.3 | 6.1 | 4 | 77.1 |
| Meta-SpikeFormer (37) | Direct Training | 55.4 | 52.4 | 4 | 80.0 |
| Meta-SpikeFormer (37) | Direct Training | 55.4 | 13.0 | 1 | 79.1 |
| MST (14) | ANN-to-SNN | 28.5 | 158.6 | 128 | 77.9 |
| **Ours (w/o ReLU)** | ANN-to-SNN | 28.7 | 3.0 | 40 | 80.0 |
| **Ours (w/ ReLU)** | ANN-to-SNN | 28.7 | 1.8 | 40 | 77.4 |

[†]For the fair comparison with the prior work, only the energy consumption for weight accumulations is included.

Table 1 compares performance between the prior work and the proposed SpikedAttention (denoted as '**Ours**' in the table). For the fair comparison, we estimate the energy consumption of only weight accumulations (discussed in Appendix A.1) for the entire timestep similar to the prior work (12; 13). Previous SNNs based on vision transformers demonstrated better performance than CNN-based SNNs (10; 11; 24). However, existing SOTA transformer-based SNNs consume $2 \sim 53\times$ higher energy than SpikedAttention (w/o ReLU) while presenting lower accuracy ($\leq 80\%$). Other SNNs feed external inputs as floating point numbers, resulting in higher energy consumption. It is noteworthy that SpikedAttention minimizes the energy consumption even with $T = 40$ by allowing only one spike per neuron. To perform a hardware-realistic energy comparison between ANNs and SNNs, we estimated the total energy consumption by using the energy metric presented in (38) which considers the data movement and the membrane potential update as well (details can be found in Appendix A.2). As a result, SpikedAttention reduces the energy consumption by 42% with only 0.6% accuracy loss when converting an ANN (w/o ReLU) which consumes 45.4mJ. Similarly, the energy consumption of SpikedAttention is 47% lower than the ANN (w/ ReLU) which consumes

Table 2: Conversion results of pre-trained models from MST (14) to SpikedAttention

| Dataset | Model | Param (M) | Energy (mJ) | Timestep | Accuracy (%) |
|---------|-------|-----------|-------------|----------|--------------|
| CIFAR-10 | ANN (w/ ReLU) | 27.5 | 42.1 | 1 | 97.5 |
| | MST (Unsigned) | 27.5 | 1089.9 | 64 | 96.3 |
| | Ours (Unsigned) | 27.5 | 21.3 | 24 | 97.3 |
| CIFAR-100 | ANN (w/ ReLU) | 27.6 | 47.1 | 1 | 87.7 |
| | MST (Unsigned) | 27.6 | 1157.1 | 64 | 85.4 |
| | Ours (Unsigned) | 27.6 | 23.7 | 24 | 86.3 |
| ImageNet | ANN (w/ ReLU) | 28.3 | 55.5 | 1 | 79.3 |
| | MST (Unsigned) | 28.3 | 1836.7 | 128 | 77.9 |
| | Ours (Unsigned) | 28.3 | 31.7 | 48 | 77.2 |

27.5 mJ. To directly compare with the previous ANN-to-SNN conversion method, i.e., MST (14), the pre-trained ANNs from MST were converted to SpikedAttention as well (Table 2). In Table 2, we computed the energy consumption of ANNs and SNNs using the hardware-realistic energy metric (Appendix A.2). Since the pre-trained models of MST embed ReLU in each layer, they were converted to SpikedAttention using unsigned neurons. The MST consumes significantly higher energy ($\geq 48\times$) than SpikedAttention because multiple spikes are generated for a longer timestep $T = 64$ or 128. In addition, the first convolution layer is based on floating point-based MACs instead of spike-based accumulations. Compared to the MST, SpikedAttention achieves higher accuracy with shorter $T$ thanks to the trace-driven matrix multiplication. As a result, we achieve SOTA SNN accuracy on ImageNet without any training or modifying the architecture while minimizing energy consumption.

## 5.2 Conversion of BERT to SpikedAttention

Table 3: Comparison between SpikedAttention and other BERT models on GLUE Benchmark

| Dataset | CoLa | MNLI | MRPC | QNLI | QQP | RTE | SST-2 | WNLI | STS-B |
|---------|------|------|------|------|-----|-----|-------|------|-------|
| **MA-BERT (39) (ANN)** | | | | | | | | | |
| **Accuracy (%)** | 59.8 | 84.7 | 84.3 | 91.4 | 91.2 | 64.6 | 92.6 | 56.3 | 84.8 |
| **Energy (mJ)** | 189.7 | 189.7 | 189.7 | 189.7 | 189.7 | 189.7 | 189.7 | 189.7 | 189.7 |
| **SpikingBERT (40) (SNN)** | | | | | | | | | |
| **Accuracy (%)** | - | 78.1 | 79.2 | 85.2 | 86.8 | 66.1 | 88.2 | - | 82.2 |
| **Ours (SNN)** | | | | | | | | | |
| **Timestep** | 24 | 24 | 16 | 24 | 24 | 16 | 16 | 16 | 24 |
| **Accuracy (%)** | 59.3 | 84.4 | 84.1 | 91.0 | 90.8 | 65.0 | 92.1 | 56.3 | 83.9 |
| **Energy (mJ)** | 81.5 | 82.1 | 77.5 | 81.6 | 82.1 | 79.1 | 79.9 | 77.7 | 79.9 |

Our proposed SpikedAttention is also applicable to NLP since it converts attention modules into spike-based computations. However, there are some functions, such as GeLU and LayerNorm, that are difficult to compute with an SNN for now. Since MA-BERT (39) replaces GELU with ReLU, converts LayerNorm to BatchNorm, and fuses normalization layers into adjacent linear layers, MA-BERT can be easily converted to SpikedAttention without any modification. Therefore, we converted MA-BERT (base-uncased) with 110M parameters to SpikedAttention. Note that accuracy loss of MA-BERT compared to traditional BERT is only 0.1%. Even though MA-BERT approximates the softmax with a two-layer neural network, we converted MA-BERT with the original softmax to SpikedAttention (thanks to WOSS). The output of the token embedding is binary coded and fed to our SNN model. By converting the pre-trained MA-BERT for text classification on GLUE benchmark (41), we observed only 0.3% accuracy loss on average without any additional training (Table 3). Note that Matthews correlation coefficient is reported for CoLA, while accuracy is reported for the other tasks. By estimating the energy consumption as presented in Appendix A.2, SpikedAttention reduces the energy by 58% on average compared to MA-BERT. The SpikedAttention is the first work that demonstrates the transformer-to-SNN conversion for NLP tasks. SpikingBERT (40) directly trains SNN as a student using knowledge distillation from a pre-trained BERT as a teacher. Compared to

SpikingBERT ($T = 125$), SpikedAttention was able to achieve 3.6% higher accuracy on average as it directly converts a well-trained BERT into an SNN without training.

## 6 Conclusion

This paper presented trace-driven matrix multiplication and winner-oriented spike shift to convert the attention module with spike-based computations. Our proposed methods accurately approximate all activations in the self-attention module as spikes without performing expensive exponent computations. Thus, our work outperforms previous transformer-based SNNs in accuracy and consumes much less energy (25.6mJ for vision tasks and 80.2mJ for language tasks on average) without structural modifications or direct training. By converting ANNs with a high number of multiplications into addition-only SNNs without any training iterations, our work makes AI more accessible on energy-constrained devices. Since the presented conversion method does not require any additional training, we can obtain SNNs without increasing the amount of carbon emissions. However, there are some limitations: first, the timestep required for SpikedAttention to maintain high accuracy is longer than directly trained SNNs. Thus, our next goal would be reducing the timestep by learning the per-layer base for better one-spike phase coding. The second limitation is that SpikedAttention do not support GeLU and LayerNorm making it difficult to be generalized to any language models. Nevertheless, our work is an essential step towards converting large language models to SNNs, and converting LayerNorm and GeLU to spike-based computations remains as our future work.

## Acknowledgments and Disclosure of Funding

This work was partially supported by the Institute of Information & Communications Technology Planning & Evaluation (IITP) funded by the Ministry of Science and ICT under Grant 2022-0-01170 and Grant RS-2023-00229849; in part by the National Research Foundation of Korea (NRF) funded by the Ministry of Science and ICT under Grant RS-2023-00258227.

## References

[1] J. Devlin, M.-W. Chang, K. Lee, and K. Toutanova, "BERT: Pre-training of deep bidirectional transformers for language understanding," in *Proceedings of the Conference of the North American Chapter of the Association for Computational Linguistics: Human Language Technologies, Volume 1 (NAACL-HLT)*, pp. 4171–4186, 2019.

[2] S. Gunasekar, Y. Zhang, J. Aneja, C. C. T. Mendes, A. Del Giorno, S. Gopi, M. Javaheripi, P. Kauffmann, G. de Rosa, O. Saarikivi, *et al.*, "Textbooks are all you need," *arXiv:2306.11644*, 2023.

[3] H. Wu, H. Zhou, M. Long, and J. Wang, "Interpretable weather forecasting for worldwide stations with a unified deep model," *Nature Machine Intelligence*, pp. 1–10, 2023.

[4] W. Peebles and S. Xie, "Scalable diffusion models with transformers," in *Proceedings of the IEEE/CVF International Conference on Computer Vision (ICCV)*, pp. 4195–4205, 2023.

[5] A. Vaswani, N. Shazeer, N. Parmar, J. Uszkoreit, L. Jones, A. N. Gomez, Ł. Kaiser, and I. Polosukhin, "Attention is all you need," in *Proceedings of the International Conference on Neural Information Processing Systems (NeurIPS)*, 2017.

[6] Z. Liu, Y. Lin, Y. Cao, H. Hu, Y. Wei, Z. Zhang, S. Lin, and B. Guo, "Swin Transformer: Hierarchical vision transformer using shifted windows," in *Proceedings of the IEEE/CVF International Conference on Computer Vision (ICCV)*, 2021.

[7] A. Radford, J. W. Kim, C. Hallacy, A. Ramesh, G. Goh, S. Agarwal, G. Sastry, A. Askell, P. Mishkin, J. Clark, *et al.*, "Learning transferable visual models from natural language supervision," in *Proceedings of International Conference on Machine Learning (ICML)*, pp. 8748–8763, PMLR, 2021.

[8] T. B. Brown, B. Mann, N. Ryder, M. Subbiah, J. Kaplan, P. Dhariwal, A. Neelakantan, P. Shyam, G. Sastry, A. Askell, *et al.*, "Language models are few-shot learners," in *Proceedings of the International Conference on Neural Information Processing Systems (NeurIPS)*, pp. 1877–1901, 2020.

[9] M. Horowitz, "1.1 Computing's energy problem (and what we can do about it)," in *Proceedings of IEEE International Solid-State Circuits Conference (ISSCC)*, 2014.

[10] B. Han, G. Srinivasan, and K. Roy, "RMP-SNN: Residual membrane potential neuron for enabling deeper high-accuracy and low-latency spiking neural network," in *Proceedings of IEEE/CVF Conference on Computer Vision and Pattern Recognition (CVPR)*, pp. 13555–13564, 2020.

[11] B. Han and K. Roy, "Deep spiking neural network: Energy efficiency through time based coding," in *Proceedings of European Conference on Computer Vision (ECCV)*, pp. 388–404, Springer, 2020.

[12] Z. Zhou, Y. Zhu, C. He, Y. Wang, S. YAN, Y. Tian, and L. Yuan, "Spikformer: When spiking neural network meets transformer," in *Proceedings of International Conference on Learning Representations (ICLR)*, 2023.

[13] M. Yao, J. Hu, Z. Zhou, L. Yuan, Y. Tian, X. Bo, and G. Li, "Spike-driven transformer," in *Proceedings of the International Conference on Neural Information Processing Systems (NeurIPS)*, 2023.

[14] Z. Wang, Y. Fang, J. Cao, Q. Zhang, Z. Wang, and R. Xu, "Masked spiking transformer," in *Proceedings of the IEEE/CVF International Conference on Computer Vision (ICCV)*, 2023.

[15] J. Deng, W. Dong, R. Socher, L.-J. Li, K. Li, and L. Fei-Fei, "Imagenet: A large-scale hierarchical image database," in *Proceedings of IEEE/CVF Conference on Computer Vision and Pattern Recognition (CVPR)*, pp. 248–255, Ieee, 2009.

[16] S. Kim, A. Gholami, Z. Yao, M. W. Mahoney, and K. Keutzer, "I-BERT: Integer-only BERT quantization," in *Proceedings of International Conference on Machine Learning (ICML)*, pp. 5506–5518, PMLR, 2021.

[17] Y. Lin, T. Zhang, P. Sun, Z. Li, and S. Zhou, "FQ-ViT: Post-training quantization for fully quantized vision transformer," in *Proceedings of International Joint Conference on Artificial Intelligence (IJCAI)*, pp. 1173–1179, 2022.

[18] H. Zheng, Y. Wu, L. Deng, Y. Hu, and G. Li, "Going deeper with directly-trained larger spiking neural networks," in *Proceedings of the AAAI Conference on Artificial Intelligence (AAAI)*, pp. 11062–11070, 2021.

[19] C. Lv, T. Li, J. Xu, C. Gu, Z. Ling, C. Zhang, X. Zheng, and X. Huang, "Spikebert: A language spikformer trained with two-stage knowledge distillation from bert," *arXiv preprint arXiv:2308.15122*, 2023.

[20] R.-J. Zhu, Q. Zhao, and J. K. Eshraghian, "Spikegpt: Generative pre-trained language model with spiking neural networks," *arXiv preprint arXiv:2302.13939*, 2023.

[21] A. L. Hodgkin and A. F. Huxley, "Currents carried by sodium and potassium ions through the membrane of the giant axon of loligo," *The Journal of Physiology*, vol. 116, no. 4, pp. 449–472, 1952.

[22] E. M. Izhikevich, "Which model to use for cortical spiking neurons?," *IEEE Transactions on Neural Networks (TNN)*, vol. 15, pp. 1063–1070, Sept. 2004.

[23] N. Fourcaud-Trocmé, D. Hansel, C. van Vreeswijk, and N. Brunel, "How spike generation mechanisms determine the neuronal response to fluctuating inputs," *The Journal of Neuroscience*, vol. 23, no. 37, pp. 11628–11640, 2003.

[24] T. Bu, W. Fang, J. Ding, P. Dai, Z. Yu, and T. Huang, "Optimal ANN-SNN conversion for high-accuracy and ultra-low-latency spiking neural networks," in *Proceedings of International Conference on Learning Representations (ICLR)*, 2021.

[25] P. U. Diehl and M. Cook, "Unsupervised learning of digit recognition using spike-timing-dependent plasticity," *Frontiers in Computational Neuroscience*, vol. 9, p. 99, Aug. 2015.

[26] B. Rueckauer and S.-C. Liu, "Conversion of analog to spiking neural networks using sparse temporal coding," in *Proceedings of IEEE International Symposium on Circuits and Systems (ISCAS)*, pp. 1–5, 2018.

[27] S. Park, S. Kim, B. Na, and S. Yoon, "T2FSNN: deep spiking neural networks with time-to-first-spike coding," in *Proceedings of ACM/IEEE Design Automation Conference (DAC)*, pp. 1–6, 2020.

[28] J. Kim, H. Kim, S. Huh, J. Lee, and K. Choi, "Deep neural networks with weighted spikes," *Neurocomputing*, vol. 311, pp. 373–386, 2018.

[29] M. Zhang, Z. Gu, N. Zheng, D. Ma, and G. Pan, "Efficient spiking neural networks with logarithmic temporal coding," *IEEE Access*, vol. 8, pp. 98156–98167, 2020.

[30] S. Hwang and J. Kung, "One-spike snn: Single-spike phase coding with base manipulation for ann-to-snn conversion loss minimization," *IEEE Transactions on Emerging Topics in Computing*, 2024.

[31] J.-P. Pfister and W. Gerstner, "Triplets of spikes in a model of spike timing-dependent plasticity," *Journal of Neuroscience*, vol. 26, Sept. 2006.

[32] M. Davies, N. Srinivasa, T.-H. Lin, G. Chinya, Y. Cao, S. H. Choday, G. Dimou, P. Joshi, N. Imam, S. Jain, *et al.*, "Loihi: A neuromorphic manycore processor with on-chip learning," *IEEE Micro*, vol. 38, no. 1, pp. 82–99, 2018.

[33] R. Hu, B. Tian, S. Yin, and S. Wei, "Efficient hardware architecture of softmax layer in deep neural network," in *Proceedings of International Conference on Digital Signal Processing (DSP)*, pp. 1–5, IEEE, 2018.

[34] W. Fang, Y. Chen, J. Ding, Z. Yu, T. Masquelier, D. Chen, L. Huang, H. Zhou, G. Li, and Y. Tian, "Spikingjelly: An open-source machine learning infrastructure platform for spike-based intelligence," *Science Advances*, vol. 9, no. 40, p. eadi1480, 2023.

[35] Z. Yao, Y. Cao, Y. Lin, Z. Liu, Z. Zhang, and H. Hu, "Leveraging batch normalization for vision transformers," in *Proceedings of the IEEE/CVF International Conference on Computer Vision Workshops (ICCVW)*, pp. 413–422, 2021.

[36] B. Rueckauer, I.-A. Lungu, Y. Hu, M. Pfeiffer, and S.-C. Liu, "Conversion of continuous-valued deep networks to efficient event-driven networks for image classification," *Frontiers in Neuroscience*, vol. 11, p. 682, 2017.

[37] M. Yao, J. Hu, T. Hu, Y. Xu, Z. Zhou, Y. Tian, B. XU, and G. Li, "Spike-driven Transformer V2: Meta spiking neural network architecture inspiring the design of next-generation neuromorphic chips," in *Proceedings of International Conference on Learning Representations (ICLR)*, 2024.

[38] F. Ottati, C. Gao, Q. Chen, G. Brignone, M. R. Casu, J. K. Eshraghian, and L. Lavagno, "To spike or not to spike: A digital hardware perspective on deep learning acceleration," *IEEE Journal on Emerging and Selected Topics in Circuits and Systems (JETCAS)*, 2023.

[39] N. W. Ming, Z. Wang, C. Liu, R. S. M. Goh, and T. Luo, "Ma-bert: Towards matrix arithmetic-only bert inference by eliminating complex non-linear functions," in *Proceedings of International Conference on Learning Representations (ICLR)*, 2022.

[40] M. Bal and A. Sengupta, "Spikingbert: Distilling bert to train spiking language models using implicit differentiation," in *Proceedings of the AAAI Conference on Artificial Intelligence (AAAI)*, vol. 38, pp. 10998–11006, 2024.

[41] A. Wang, A. Singh, J. Michael, F. Hill, O. Levy, and S. Bowman, "Glue: A multi-task benchmark and analysis platform for natural language understanding," in *Proceedings of International Conference on Learning Representations (ICLR)*, 2018.

[42] R. Yin, A. Moitra, A. Bhattacharjee, Y. Kim, and P. Panda, "Sata: Sparsity-aware training accelerator for spiking neural networks," *IEEE Transactions on Computer-Aided Design of Integrated Circuits and Systems (TCAD)*, 2022.

[43] T. Ajayi and D. Blaauw, "Openroad: Toward a self-driving, open-source digital layout implementation tool chain," in *Proceedings of Government Microcircuit Applications and Critical Technology Conference (GOMACTech)*, 2019.

[44] H. Wang, Z. Wang, M. Du, F. Yang, Z. Zhang, S. Ding, P. Mardziel, and X. Hu, "Score-CAM: Score-weighted visual explanations for convolutional neural networks," in *Proceedings of IEEE/CVF Conference on Computer Vision and Pattern Recognition Workshops (CVPRW)*, pp. 111–119, 2020.

[45] P. Diehl, D. Neil, J. Binas, M. Cook, S. Liu, and M. Pfeiffer, "Fast-classifying, high-accuracy spiking deep networks through weight and threshold balancing," in *Proceedings of International Joint Conference on Neural Networks (IJCNN)*, 2015.

[46] S. Kim, S. Park, B. Na, and S. Yoon, "Spiking-yolo: spiking neural network for energy-efficient object detection," in *Proceedings of the AAAI Conference on Artificial Intelligence (AAAI)*, 2020.

[47] P. Rajpurkar, J. Zhang, K. Lopyrev, and P. Liang, "SQuAD: 100,000+ questions for machine comprehension of text," in *Proceedings of the the Conference on Empirical Methods in Natural Language Processing (EMNLP)* (J. Su, K. Duh, and X. Carreras, eds.), (Austin, Texas), pp. 2383–2392, Association for Computational Linguistics, Nov. 2016.

# A Energy Metrics

## A.1 Simple Energy Metric Considering Only Synaptic Operations

For the fair comparison with the prior work (12; 13), energy cost of synaptic operations is estimated based on the power consumption of 32-bit floating-point arithmetic units synthesized in 45nm CMOS technology (9). The synaptic operations of the ANN architecture are MACs, i.e., input activations are multiplied by the corresponding weights and partial sums are accumulated. We estimate the energy consumption of an ANN (denoted as $E_{ANN}$) by using the number of synaptic operations associated with the layer $l$ of the ANN ($SynOP^l$), which is defined as:

$$E_{ANN} = (E_{add} + E_{mult}) \cdot \sum_{l=1}^{N_L} SynOP^l \tag{9}$$

where $E_{add} + E_{mult}$ is $3.7pJ$ for a single 32-bit floating point MAC operation.

Unlike ANNs that require both multipliers and adders, SNNs only require additions. Only weights that are connected to pre-synaptic neurons that fire spikes are accumulated together. Therefore, the energy consumption of an SNN can be estimated by:

$$E_{SNN} = E_{add} \cdot \sum_{t=1}^{T} \sum_{l=1}^{N_L} (\gamma_t^l \cdot SynOP^l) \tag{10}$$

where $\gamma_t^l$ is the ratio of spikes at timestep $t$ and $E_{add}$ is $0.9pJ$ for a single 32-bit floating point addition. Note that the energy numbers reported in the Table 1 accounts for the entire timesteps in SNN models.

## A.2 More Realistic Energy Metric Considering Hardware

The energy metric described in Appendix A.1 only considers the computing energy. However, a large portion of energy is being consumed by repeatedly reading/writing activations and weights, as well as updating the neuron potential over multiple timesteps. The data movement highly depends on the underlying hardware architecture, e.g., on-chip memory capacity and interconnect topology. For instance, membrane potentials can be fetched from DRAM, whereas the neuron model is computed on the core (42). In that case, the energy cost of fetching membrane potentials becomes significant. On the contrary, Intel's Loihi (32) places a large on-chip memory to store membrane potentials or traces inside the chip. Therefore, to perform a hardware-agnostic energy comparison between ANNs and SNNs, we estimated the total energy consumption by using the energy metric presented in (38), which considers the energy of reading weights and spikes, the energy of computing the neuron model, and the energy of writing spikes back to the memory.

First, the energy consumption of an ANN is computed by:

$$\begin{aligned}
E_{ANN} &= E_{rd_{tot}} + E_{MAC} + E_{offmap}, \\
E_{rd_{tot}} &= 2E_{rd} \cdot \sum_{l=1}^{N_L} \gamma_l SynOP^l, \\
E_{MAC} &= (E_{add} + E_{mult}) \cdot \sum_{l=1}^{N_L} \gamma^l SynOP^l, \\
E_{offmap} &= E_{wr} \cdot \sum_{l=1}^{N_L} Act^l,
\end{aligned} \tag{11}$$

where $E_{rd_{tot}}$ is the energy for reading weights and activations, $E_{MAC}$ is the energy for arithmetic operations, and $E_{offmap}$ is the energy for writing MAC results to the scratchpad memory. The $Act^l$ is number of output activations at layer $l$ and $E_{rd}/E_{wr}$ is set to $5pJ$ for the 32-bit read/write operation (9). Also, the activation sparsity is considered by the term $\gamma_l$ for the fair comparison with the SNN.

Next, the energy consumption of an SNN is computed as follows:

$$E_{SNN} = E_{rd_{tot}} + E_{AC} + E_{Neuron} + E_{offmap},$$

$$E_{rd_{tot}} = (1 + \frac{1}{32})E_{rd} \cdot \sum_{t=1}^{T} \sum_{l=1}^{N_L} \gamma_t^l \cdot SynOP^l,$$

$$E_{MAC} = E_{add} \cdot \sum_{t=1}^{T} \sum_{l=1}^{N_L} \gamma_t^l \cdot SynOP^l,$$

$$E_{Neuron} = (E_{rd} + E_{mult} + E_{add} + E_{comp} + E_{sub} + E_{wr}) \cdot \sum_{t=1}^{T} \sum_{l=1}^{N_L} Act^l,$$

$$E_{offmap} = \frac{E_{wr}}{32} \cdot \sum_{t=1}^{T} \sum_{l=1}^{N_L} Act^l,$$

(12)

where $E_{rd_{tot}}$ is the energy for reading weights and input spikes, $E_{AC}$ is the energy for synaptic operations (i.e., additions), $E_{Neuron}$ is the energy for computing neuron models and $E_{offmap}$ is the energy for writing output spikes to the scratchpad memory. In $E_{rd_{tot}}$ and $E_{offmap}$, the energy of reading/writing 1-bit spike is $E_{rd}/32$ (or $E_{wr}/32$) since $E_{rd}/E_{wr}$ is the read/write energy for 32-bit data. The $E_{Neuron}$ consists of retrieving the neuron's potential ($E_{rd}$), multiplying leak factor ($E_{multi}$), adding input to potential ($E_{add}$), comparing with the threshold ($E_{comp}$), subtracting the potential by the threshold ($E_{sub}$), and finally writing the potential back to the memory ($E_{wr}$). As in (38), we defined $E_{sub}$ and $E_{comp}$ to have the same energy value as $E_{add} = 0.9pJ$. Note that SpikedAttention stops generating spikes at neurons that fired already implying that there is no energy consumed by those neurons afterwards. The energy overhead of computing the neuron model was 30~40% of the total energy, even though the total timestep of SpikedAttention converted from Swin-T was $T = 40$. This is because the number of synaptic operations ($\sum_{l=1}^{N_L} SynOP^l$) is larger than the number of output activations ($\sum_{l=1}^{N_L} Act^l$). Therefore, even with the overhead of computing the neuron model, SpikedAttention, which generates a single spike for the entire timestep (i.e., $\sum_{t=1}^{T} \gamma_t^l \leq 1$), is energy efficient since the sparsity of synaptic operations is the most critical factor in improving energy efficiency.

## B  LIF Model with Phase Coding

In this work, our phase coding with the base '$B$' follows the phase coding scheme presented in (28), which fixes the base to 2. The dynamics of a neuron in the phase coding can be computed by Eq. 1 by changing the leak factor of the LIF to the selected base $B$.

To express the equivalence clearly, we formulate it as the following theorem:

**Theorem B.1.** *Given the following neuron dynamics of the phase coding with the base B*

$$u_j(t) = u_j(t-1) + \sum_{i=1}^{N_{pre}} w_{ij} B^{-t} S_i(t),$$

$$S_j(t) = \begin{cases} 1, if\, u_j(t) \geq V_\theta \times B^{-t} \\ 0, otherwise \end{cases}$$

(A1)

*It can be formulated to the following LIF-based neural dynamics:*

$$v_j(t) = B \cdot v_j(t-1) + \sum_{i=1}^{N_{pre}} w_{ij} S_i(t), S_j(t) = \begin{cases} 1, if\, v_j(t) \geq V_\theta \\ 0, otherwise \end{cases}$$

(A2)

*Proof.* The equivalence of the two dynamics lies in the equivalence of the conditions, $u_j(t) \geq B^{-t} \cdot V_\theta$ and $v_j(t) \geq V_\theta$, i.e., $u_j(t) = v_j(t) \cdot B^{-t}, t = 1, \ldots, T$. We prove this using mathematical induction. For $t = 1$, we have $u_j(1) = v_j(1) \cdot B^{-1} = 0$. Assume that $u_j(t) = v_j(t) \cdot B^{-t}$ holds for $t = 1, ..., t$,

next we prove that $u_j(t+1) = v_j(t+1) \cdot B^{-(t+1)}$:

$$
\begin{aligned}
u_j(t+1) &= u_j(t) + \sum_{i=1}^{N_{pre}} w_{ij} B^{-(t+1)} S_i(t+1) \\
&= v_j(t) B^{-t} + \sum_{i=1}^{N_{pre}} w_{ij} B^{-(t+1)} S_i(t+1) \\
&= B^{-(t+1)} \left( B \cdot v_j(t) + \sum_{i=1}^{N_{pre}} w_{ij} S_i(t+1) \right) \\
&= B^{-(t+1)} \cdot v_j(t+1).
\end{aligned}
\tag{13}
$$

∎

## C    Hardware Overhead of Implementing WOSS Neurons

To implement WOSS in the neuromorphic hardware, we need some modifications to the hardware structure of the existing LIF neuron. The recent Loihi2 (32) allows users to select a neuron model among many neuron models implemented in hardware. Thus, we designed WOSS neuron in Verilog and synthesized the hardware module in 45nm CMOS technology using OpenROAD (43) to estimate the area/power overhead when realizing it on top of the commercial neuromorphic hardware. Please note that WOSS neurons are only required for the softmax layer accounting for 19% of all neurons at each attention block of Swin-T. The remaining 81% are implemented as typical LIF neurons. As summarized in Table. 4, compared to a general LIF model, the proposed WOSS neuron increases the area by 52% and the power consumption by 65%. This translates to 9.88% larger area and 12.35% higher power consumption to support the proposed WOSS method for softmax operations.

Table 4: Area and power overheads of a hardware module for supporting the WOSS neuron

| Tech Node: 45nm | Area ($\mu m^2$) | Power ($\mu W$) |
|---|---|---|
| General LIF | 807 | 542 |
| WOSS LIF | 1227 | 894 |
| Overhead (%) | 52 | 65 |

## D    Visualization of Attention Map

Fig. 6 presents the attention maps, highlighting the areas of activation's interest in the image at the outputs of several self-attention layers. The attention maps from the pre-trained Swin Transformer and the SpikedAttention were extracted using Score-CAM (44). The attention maps of SpikedAttention from the 9th to 12th attention layer are similar to those of Swin-T, even though the conversion error increases at deeper layers. This visualization of attention maps implies that our ANN-to-SNN conversion method successfully transforms attention modules into spike-based operations.

## E    Selecting the Optimal Base $B$ for a Given Timestep $T$

To achieve the optimal performance, SpikedAttention needs to set an optimal base. For selecting the optimal base, we define an error function to find the optimal base $B$, which presents the smallest difference between approximated spike values and true activations. The most common error evaluation is an absolute or relative error. However, we need something more suitable for our work. It is not effective to use absolute error like MSE , i.e., $\frac{1}{n} \sum (x - \hat{x})^2$ where $x$ is the actual value and $\hat{x}$ is the approximated value, since larger error gets accumulated for large $x$ as our spike approximation is log-based. The relative error like ARE, i.e., $\frac{1}{n} \sum |(\frac{x-\hat{x}}{x})|$, is also not a good candidate, since the same relative difference may imply different timestep scale which depends on the base $B$. For instance, for a given $x$, $1.1 \times x$ is $1.95$ timesteps away from $x$ when the base is $1.05$. When the base increases to $1.07$, the timestep difference between the two values decreases to $1.41$.

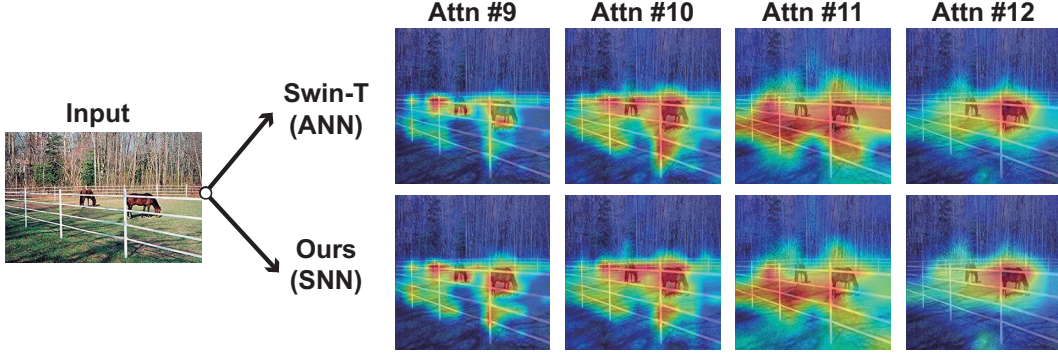

Figure 6: Comparison of attention maps based on Score-CAM (44) between Swin-T (baseline ANN) and SpikedAttention (Ours; $B = 1.16$ and $T = 40$) on ImageNet-1K at the output of various attention blocks.

We can accurately estimate the approximation error between the real-valued activation and the decoded spike value using the logarithmic error $E_q(x)$ as follows:

$$E_q(x) = \begin{cases} B \times |L+1|, & \text{if } L > -1 \\ B \times |L - \lfloor L \rfloor|, & \text{if } -1 \geq L \geq -T \\ B \times 0.5, & \text{otherwise.} \end{cases} \quad (14)$$

where $x$ is the sampled activation from $\mathbf{X}$ of ANN and '$L = \log_B x - \log_B (\max(\mathbf{X}))$' is the logarithm of normalized activation. Note that as the weights are normalized (45; 46) by $\max(\mathbf{X})$, i.e., the maximum value of $\mathbf{X}$ in a subset of the training dataset, each activation is approximated by a spike as '$\max(\mathbf{X}) \times B^{-t}$'. In other words, spike time $t$ represents $-\lfloor L \rfloor$. Error with values less than '$\max(\mathbf{X}) \times B^{-T}$' is fixed to 0.5, since $\log_B 0$ is not defined.

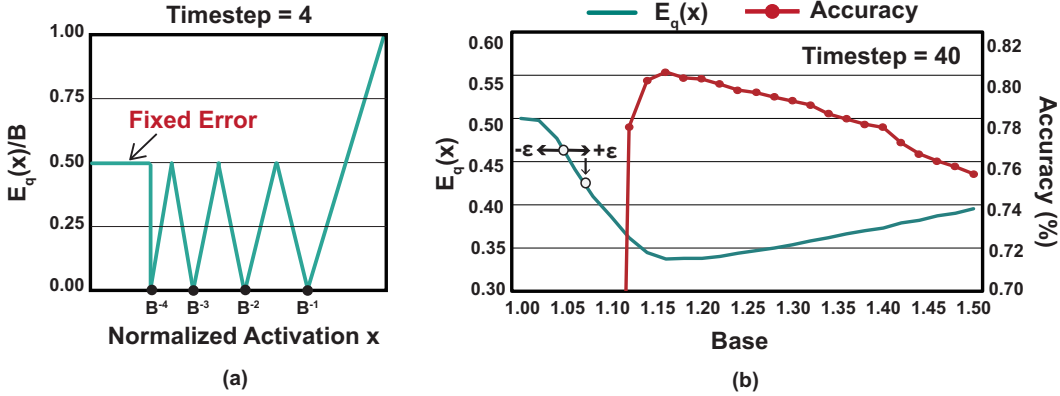

Figure 7: (a) Normalized logarithmic error ($\frac{E_q(x)}{B}$) with respect to the normalized activation ($\frac{x}{\max(\mathbf{X})}$) at a given timestep $T = 4$. (b) Logarithmic error ($E_q(x)$) and the accuracy on the ImageNet classification task at various base $B$ values at a given timestep $T = 40$.

Fig. 7(a) shows the error function for a given timestep $T = 4$. The Eq. (14) first computes the $\log_B$-based absolute error, i.e., $|L - \lfloor L \rfloor|$, where $L$ is '$\log_B x - \log_B (\max(\mathbf{X}))$'. By doing this, we can accurately measure the error when approximating an activation with a spike having $B^{-t}$ value. Two exception ranges are (i) $L > -1$ and (ii) $L < -T$ and they are correctly handled to estimate the error conservatively. Then, this $\log_B$-based error term is multiplied by $B$ to set the timestep scale properly.

To find the optimal base, our method sets the initial base $B_0$ in the midpoint of the search space $(1.0, 1.5]$. Then, we compute the $\sum E_q(x)$ for both $B = B_0 - \epsilon$ and $B = B_0 + \epsilon$, and

update the base to one of them that provides a smaller error. The optimal base with the smallest error can be found within a given search space by iterating the process several times. Fig. 7(b) presents the logarithmic error $E_q(x)$ and the accuracy according to the base $B$. The accuracy loss follows the same trend to our logarithmic error function, i.e., as $B$ gets smaller, the accuracy increases while our error term decreases. If $B$ gets too small, however, the accuracy decreases (equivalently, our error term increases) due to the increased number of underflows. As a result, we utilized the $\log_B$-based absolute error multiplied by the base when determining the optimal base for a given benchmark.

## F  Performance on Question Answering

Table 5: Performance of SpikedAttention on Question Answering

| Question Answering | | | | | |
|---|---|---|---|---|---|
| Dataset | Model | Param (M) | Energy (mJ) | Timestep | F1 Score (%) |
| SQuAD | MA-BERT (w/o ReLU) | 110 | 833.8 | 1 | 88.3 |
| | Ours (Signed) | 110 | 337.9 | 24 | 87.2 |

To demonstrate that SpikedAttention can reduce energy consumption for more complex tasks, we converted MA-BERT for question answering to SpikedAttention. We trained the existing MA-BERT on the SQuAD dataset (47) for question answering and converted it to spike-based computations. As a result, SpikedAttention achieves an energy reduction of 59.5% with only 1.1% accuracy loss, as presented on Table 5.

## G  Impact of Total Timestep on Accuracy and Energy Consumption

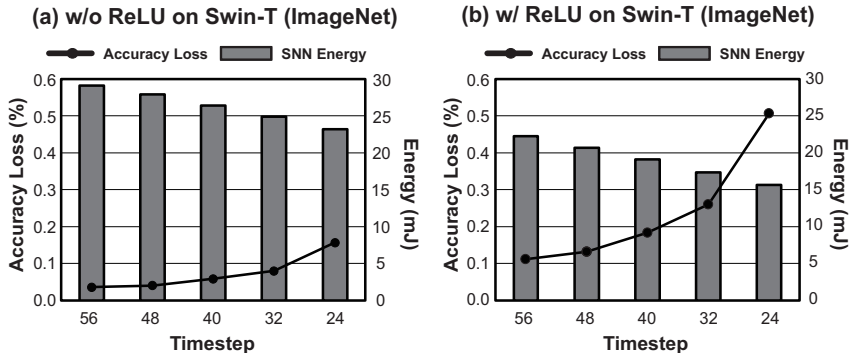

Figure 8: Accuracy loss and the energy consumption at various timesteps when converting Swin-T to an SNN for ImageNet classification. (a) SpikedAttention from Swin-T without any modification, and (b) SpikedAttention from Swin-T with inserted ReLUs.

Regarding the accuracy, as discussed in Appendix E, the larger the timestep T, the smaller the base value of an one-spike SNN can be. A smaller base reduces the conversion loss, leading to higher accuracy. Regarding the energy consumption, as discussed in Appendix A.2, a longer timestep incurs higher energy consumption due to more neuron model computations and data movements. To summarize, a longer timestep increases the energy consumption while reducing the accuracy loss. Fig. 8 shows the accuracy loss and the energy consumption at various timesteps when converting Swin-T to an SNN for ImageNet classification. It clearly shows the trade-off between the total timestep and the accuracy/energy consumption. Note that even at $T = 24$ the accuracy loss is less than 0.5%.

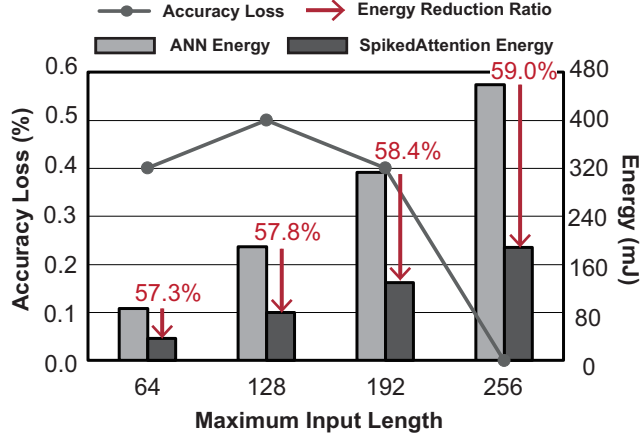

Figure 9: Energy consumption of MA-BERT and SpikedAttention, and accuracy loss of BERT-to-SNN conversion on the GLUE SST2 dataset according to the maximum input length (i.e., 64, 128, 192 and 256).

## H    Energy Consumption at Various Input Token Lengths

Energy consumption of ANN/SNN varies with the input token length for NLP tasks. To evalute this, we varied the input token length of MA-BERT (i.e., target ANN) from 64 to 256 on SST-2 dataset. Also, the same lengths of input tokens are fed into the converted SpikedAttention model for the energy evaluation. Fig. 9 shows energy consumptions at various input lengths for both ANN and SpikedAttention. In addition, the accuracy losses due to the ANN-to-SNN conversion are presented which are negligible (< 1%).

For both ANN and SpikedAttention, the energy consumption increases as the maximum input length increases from 64 to 256. This is because the number of computations increases in the attention module with the increase in the input length. For instance, MA-BERT with the input length of 128 consumes 189.7mJ of energy, while MA-BERT with the input length of 256 consumes $458.9mJ$ ($2.4\times$). Since SpikedAttention benefits from fully spiked-based computations, the energy consumption for input lengths of 128 and 256 is $79.9mJ$ and $188.0mJ$, respectively. These energy numbers imply that SpikedAttention is $2.4\times$ more energy-efficient compared to the ANN, regardless of the input token length. As we optimized the spiked-based computation in the attention module with WOSS and trace-based matrix multiplication, the energy reduction ratio of SpikedAttention compared to ANN slightly increases as the input length increases.

